# Inferring Motor Programs from Images of Handwritten Digits

**Geoffrey Hinton and Vinod Nair**
Department of Computer Science, University of Toronto
10 King's College Road, Toronto, M5S 3G5 Canada
{*hinton,vnair*}*@cs.toronto.edu*

## Abstract

We describe a generative model for handwritten digits that uses two pairs of opposing springs whose stiffnesses are controlled by a motor program. We show how neural networks can be trained to infer the motor programs required to accurately reconstruct the MNIST digits. The inferred motor programs can be used directly for digit classification, but they can also be used in other ways. By adding noise to the motor program inferred from an MNIST image we can generate a large set of very different images of the same class, thus enlarging the training set available to other methods. We can also use the motor programs as additional, highly informative outputs which reduce overfitting when training a feed-forward classifier.

## 1 Overview

The idea that patterns can be recognized by figuring out how they were generated has been around for at least half a century [1, 2] and one of the first proposed applications was the recognition of handwriting using a generative model that involved pairs of opposing springs [3, 4]. The "analysis-by-synthesis" approach is attractive because the true generative model should provide the most natural way to characterize a class of patterns. The handwritten 2's in figure 1, for example, are very variable when viewed as pixels but they have very similar motor programs. Despite its obvious merits, analysis-by-synthesis has had few successes, partly because it is computationally expensive to invert non-linear generative models and partly because the underlying parameters of the generative model are unknown for most large data sets. For example, the only source of information about how the MNIST digits were drawn is the images themselves.

We describe a simple generative model in which a pen is controlled by two pairs of opposing springs whose stiffnesses are specified by a motor program. If the sequence of stiffnesses is specified correctly, the model can produce images which look very like the MNIST digits. Using a separate network for each digit class, we show that backpropagation can be used to learn a "recognition" network that maps images to the motor programs required to produce them. An interesting aspect of this learning is that the network creates its own training data, so it does not require the training images to be labelled with motor programs. Each recognition network starts with a single example of a motor program and grows an "island of competence" around this example, progressively extending the region over which it can map small changes in the image to the corresponding small changes in the motor program (see figure 2).

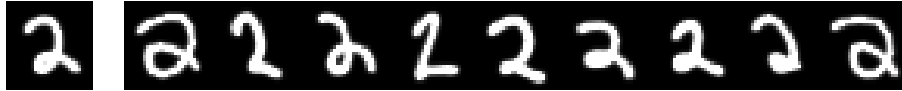

Figure 1: An MNIST image of a 2 and the additional images that can be generated by inferring the motor program and then adding random noise to it. The pixels are very different, but they are all clearly twos.

Fairly good digit recognition can be achieved by using the 10 recognition networks to find 10 motor programs for a test image and then scoring each motor program by its squared error in reconstructing the image. The 10 scores are then fed into a softmax classifier. Recognition can be improved by using PCA to model the distribution of motor trajectories for each class and using the distance of a motor trajectory from the relevant PCA hyperplane as an additional score.

Each recognition network is solving a difficult global search problem in which the correct motor program must be found by a single, "open-loop" pass through the network. More accurate recognition can be achieved by using this open-loop global search to initialize an iterative, closed-loop local search which uses the error in the reconstructed image to revise the motor program. This requires reconstruction errors in pixel space to be mapped to corrections in the space of spring stiffnesses. We cannot backpropagate errors through the generative model because it is just a hand-coded computer program. So we learn "generative" networks, one per digit class, that emulate the generator. After learning, backpropagation through these generative networks is used to convert pixel reconstruction errors into stiffness corrections.

Our final system gives 1.82% error on the MNIST test set which is similar to the 1.7% achieved by a very different generative approach [5] but worse than the 1.53% produced by the best backpropagation networks or the 1.4% produced by support vector machines [6]. It is much worse than the 0.4% produced by convolutional neural networks that use cleverly enhanced training sets [7]. Recognition of test images is quite slow because it uses ten different recognition networks followed by iterative local search. There is, however, a much more efficient way to make use of our ability to extract motor programs. They can be treated as additional output labels when using backpropagation to train a single, multi-layer, discriminative neural network. These additional labels act as a very informative regularizer that reduces the error rate from 1.53% to 1.27% in a network with two hidden layers of 500 units each. This is a new method of improving performance that can be used in conjunction with other tricks such as preprocessing the images, enhancing the training set or using convolutional neural nets [8, 7].

## 2    A simple generative model for drawing digits

The generative model uses two pairs of opposing springs at right angles. One end of each spring is attached to a frictionless horizontal or vertical rail that is 39 pixels from the center of the image. The other end is attached to a "pen" that has significant mass. The springs themselves are weightless and have zero rest length. The pen starts at the equilibrium position defined by the initial stiffnesses of the four springs. It then follows a trajectory that is determined by the stiffness of each spring at each of the 16 subsequent time steps in the motor program. The mass is large compared with the rate at which the stiffnesses change, so the system is typically far from equilibrium as it follows the smooth trajectory. On each time step, the momentum is multiplied by 0.9 to simulate viscosity. A coarse-grain trajectory is computed by using one step of forward integration for each time step in the motor program, so it contains 17 points. The code is at www.cs.toronto.edu/$\sim$ hinton/code.

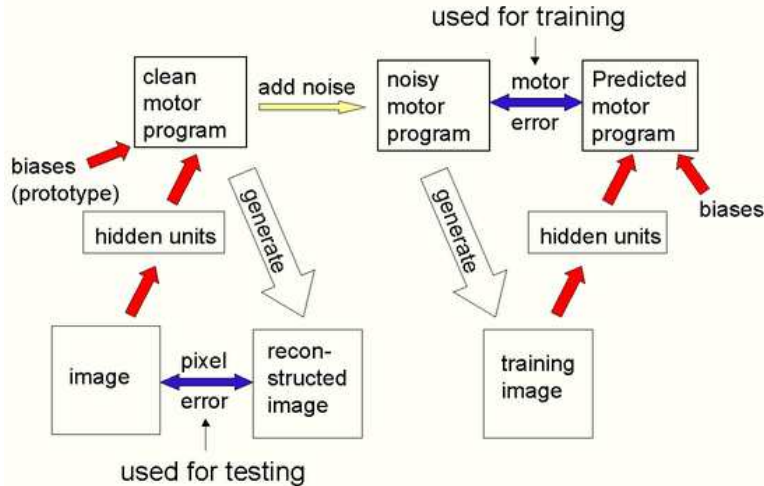

Figure 2: The training data for each class-specific recognition network is produced by adding noise to motor programs that are inferred from MNIST images using the current parameters of the recognition network. To initiate this process, the biases of the output units are set by hand so that they represent a prototypical motor program for the class.

Given a coarse-grain trajectory, we need a way of assigning an intensity to each pixel. We tried various methods until we hand-evolved one that was able to reproduce the MNIST images fairly accurately, but we suspect that many other methods would be just as good. For each point on the coarse trajectory, we share two units of ink between the the four closest pixels using bilinear interpolation. We also use linear interpolation to add three fine-grain trajectory points between every pair of coarse-grain points. These fine-grain points also contribute ink to the pixels using bilinear interpolation, but the amount of ink they contribute is zero if they are less than one pixel apart and rises linearly to the same amount as the coarse-grain points if they are more than two pixels apart. This generates a thin skeleton with a fairly uniform ink density. To flesh-out the skeleton, we use two "ink parameters", $a, b$, to specify a $3 \times 3$ kernel of the form $b(1+a)[\frac{a}{12}, \frac{a}{6}, \frac{a}{12}; \quad \frac{a}{6}, 1-a, \frac{a}{6}; \quad \frac{a}{12}, \frac{a}{6}, \frac{a}{12}]$ which is convolved with the image four times. Finally, the pixel intensities are clipped to lie in the interval [0,1]. The matlab code is at www.cs.toronto.edu/$\sim$ hinton/code. The values of $2a$ and $b/1.5$ are additional, logistic outputs of the recognition networks[1].

## 3  Training the recognition networks

The obvious way to learn a recognition network is to use a training set in which the inputs are images and the target outputs are the motor programs that were used to generate those images. If we knew the distribution over motor programs for a given digit class, we could easily produce such a set by running the generator. Unfortunately, the distribution over motor programs is exactly what we want to learn from the data, so we need a way to train

the recognition network without knowing this distribution in advance. Generating scribbles from random motor programs will not work because the capacity of the network will be wasted on irrelevant images that are far from the real data.

Figure 2 shows how a single, prototype motor program can be used to initialize a learning process that creates its own training data. The prototype consists of a sequence of $4 \times 17$ spring stiffnesses that are used to set the biases on $68$ of the $70$ logistic output units of the recognition net. If the weights coming from the $400$ hidden units are initially very small, the recognition net will then output a motor program that is a close approximation to the prototype, whatever the input image. Some random noise is then added to this motor program and it is used to generate a training image. So initially, all of the generated training images are very similar to the one produced by the prototype. The recognition net will therefore devote its capacity to modeling the way in which small changes in these images map to small changes in the motor program. Images in the MNIST training set that are close to the prototype will then be given their correct motor programs. This will tend to stretch the distribution of motor programs produced by the network along the directions that correspond to the manifold on which the digits lie. As time goes by, the generated training set will expand along the manifold for that digit class until all of the MNIST training images of that class are well modelled by the recognition network.

It takes about $10$ hours in matlab on a 3 GHz Xeon to train each recognition network. We use minibatches of size $100$, momentum of $0.9$, and adaptive learning rates on each connection that increase additively when the sign of the gradient agrees with the sign of the previous weight change and decrease multiplicatively when the signs disagree [9]. The net is generating its own training data, so the objective function is always changing which makes it inadvisable to use optimization methods that go as far as they can in a carefully chosen direction. Figures 3 and 4 show some examples of how well the recognition nets perform after training. Nearly all models achieve an average squared pixel error of less than 15 per image on their validation set (pixel intensities are between $0$ and $1$ with a preponderance of extreme values). The inferred motor programs are clearly good enough to capture the diverse handwriting styles in the data. They are not good enough, however, to give classification performance comparable to the state-of-the-art on the MNIST database. So we added a series of enhancements to the basic system to improve the classification accuracy.

## 4 Enhancements to the basic system

**Extra strokes in ones and sevens.** One limitation of the basic system is that it draws digits using only a single stroke (i.e. the trajectory is a single, unbroken curve). But when people draw digits, they often add extra strokes to them. Two of the most common examples are the dash at the bottom of ones, and the dash through the middle of sevens (see examples in figure 5). About 2.2% of ones and 13% of sevens in the MNIST training set are dashed and not modelling the dashes reduces classification accuracy significantly. We model dashed ones and sevens by augmenting their basic motor programs with another motor program to draw the dash. For example, a dashed seven is generated by first drawing an ordinary seven using the motor program computed by the seven model, and then drawing the dash with a motor program computed by a separate neural network that models only dashes.

Dashes in ones and sevens are modeled with two different networks. Their training proceeds the same way as with the other models, except now there are only $50$ hidden units and the training set contains only the dashed cases of the digit. (Separating these cases from the rest of the MNIST training set is easy because they can be quickly spotted by looking at the difference between the images and their reconstructions by the dashless digit model.) The net takes the entire image of a digit as input, and computes the motor program for just the dash. When reconstructing an unlabelled image as say, a seven, we compute both

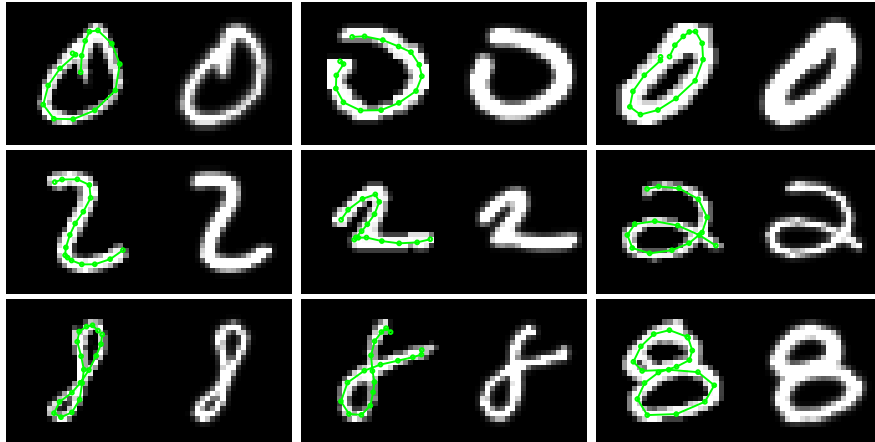

Figure 3: Examples of validation set images reconstructed by their corresponding model. In each case the original image is on the left and the reconstruction is on the right. Superimposed on the original image is the pen trajectory.

the dashed and dashless versions of seven and pick the one with the lower squared pixel error to be that image's reconstruction as a seven. Figure 5 shows examples of images reconstructed using the extra stroke.

**Local search.** When reconstructing an image in its own class, a digit model often produces a sensible, overall approximation of the image. However, some of the finer details of the reconstruction may be slightly wrong and need to be fixed up by an iterative local search that adjusts the motor program to reduce the reconstruction error. We first approximate the graphics model with a neural network that contains a single hidden layer of 500 logistic units. We train one such generative network for each of the ten digits and for the dashed version of ones and sevens (for a total of 12 nets). The motor programs used for training are obtained by adding noise to the motor programs inferred from the training data by the relevant, fully trained recognition network. The images produced from these motor programs by the graphics model are used as the targets for the supervised learning of each generative network. Given these targets, the weight updates are computed in the same way as for the recognition networks.

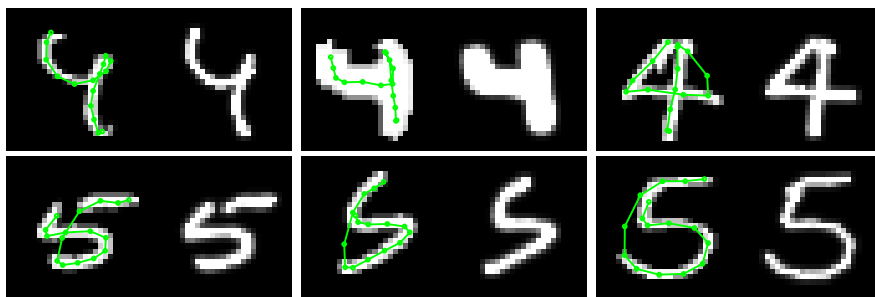

Figure 4: To model 4's we use a single smooth trajectory, but turn off the ink for timesteps 9 and 10. For images in which the pen does not need to leave the paper, the recognition net finds a trajectory in which points 8 and 11 are close together so that points 9 and 10 are not needed. For 5's we leave the top until last and turn off the ink for timesteps 13 and 14.

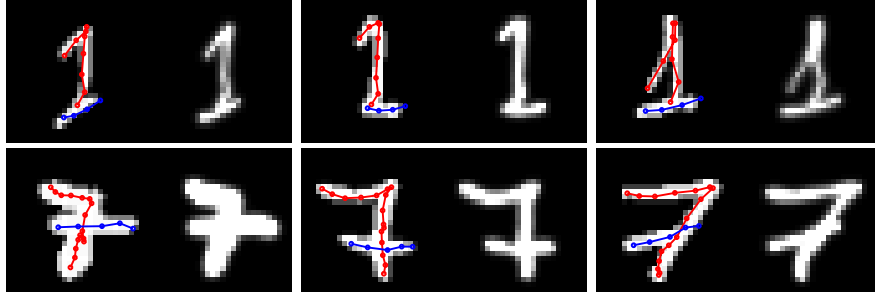

Figure 5: Examples of dashed ones and sevens reconstructed using a second stroke. The pen trajectory for the dash is shown in blue, superimposed on the original image.

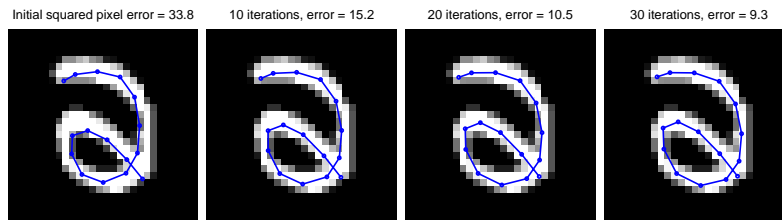

Figure 6: An example of how local search improves the detailed registration of the trajectory found by the correct model. After 30 iterations, the squared pixel error is less than a third of its initial value.

Once the generative network is trained, we can use it to iteratively improve the initial motor program computed by the recognition network for an image. The main steps in one iteration are: 1) compute the error between the image and the reconstruction generated from the current motor program by the graphics model; 2) backpropagate the reconstruction error through the generative network to calculate its gradient with respect to the motor program; 3) compute a new motor program by taking a step along the direction of steepest descent plus $0.5$ times the previous step. Figure 6 shows an example of how local search improves the reconstruction by the correct model. Local search is usually less effective at improving the fits of the wrong models, so it eliminates about 20% of the classification errors on the validation set.

**PCA model of the image residuals.** The sum of squared pixel errors is not the best way of comparing an image with its reconstruction, because it treats the residual pixel errors as independent and zero-mean Gaussian distributed, which they are not. By modelling the structure in the residual vectors, we can get a better estimate of the conditional probability of the image given the motor program. For each digit class, we construct a PCA model of the image residual vectors for the training images. Then, given a test image, we project the image residual vector produced by each inferred motor program onto the relevant PCA hyperplane and compute the squared distance between the residual and its projection. This gives ten scores for the image that measure the quality of its reconstructions by the digit models. We don't discard the old sum of squared pixel errors as they are still useful for classifying most images correctly. Instead, all twenty scores are used as inputs to the classifier, which decides how to combine both types of scores to achieve high classification accuracy.

**PCA model of trajectories.** Classifying an image by comparing its reconstruction errors for the different digit models tacitly relies on the assumption that the incorrect models will reconstruct the image poorly. Since the models have only been trained on images in their

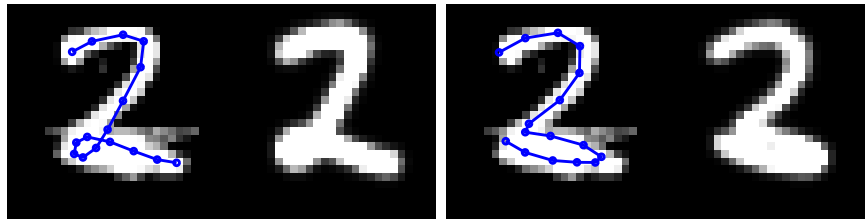

Figure 7: Reconstruction of a two image by the two model (left box) and by the three model (right box), with the pen trajectory superimposed on the original image. The three model sharply bends the bottom of its trajectory to better explain the ink, but the trajectory prior for three penalizes it with a high score. The two model has a higher squared error, but a much lower prior score, which allows the classifier to correctly label the image.

own class, they often do reconstruct images from other classes poorly, but occasionally they fit an image from another class well. For example, figure 7 shows how the three model reconstructs a two image better than the two model by generating a highly contorted three. This problem becomes even more pronounced with local search which sometimes contorts the wrong model to fit the image really well. The solution is to learn a PCA model of the trajectories that a digit model infers from images in its own class. Given a test image, the trajectory computed by each digit model is scored by its squared distance from the relevant PCA hyperplane. These 10 "prior" scores are then given to the classifier along with the 20 "likelihood" scores described above. The prior scores eliminate many classification mistakes such as the one in figure 7.

## 5   Classification results

To classify a test image, we apply multinomial logistic regression to the 30 scores – *i.e.* we use a neural network with no hidden units, 10 softmax output units and a cross-entropy error. The net is trained by gradient descent using the scores for the validation set images. To illustrate the gain in classification accuracy achieved by the enhancements explained above, table 1 gives the percent error on the validation set as each enhancement is added to the system. Together, the enhancements almost halve the number of mistakes.

| Enhancements | Validation set % error | Test set % error |
|:---:|:---:|:---:|
| None | 4.43 | |
| 1 | 3.84 | |
| 1, 2 | 3.01 | |
| 1, 2, 3 | 2.67 | |
| 1, 2, 3, 4 | 2.28 | 1.82 |

Table 1: The gain in classification accuracy on the validation set as the following enhancements are added: 1) extra stroke for dashed ones and sevens, 2) local search, 3) PCA model of image residual, and 4) PCA trajectory prior. To avoid using the test set for model selection, the performance on the official test set was only measured for the final system.

## 6   Discussion

After training a single neural network to output both the class label and the motor program for all classes (as described in section 1) we tried ignoring the label output and classifying

the test images by using the cost, under 10 different PCA models, of the trajectory defined by the inferred motor program. Each PCA model was fitted to the trajectories extracted from the training images for a given class. This gave 1.80% errors which is as good as the 1.82% we got using the 10 separate recognition networks and local search. This is quite surprising because the motor programs produced by the single network were simplified to make them all have the same dimensionality and they produced significantly poorer reconstructions. By only using the 10 digit-specific recognition nets to create the motor programs for the *training* data, we get much faster recognition of test data because at test time we can use a single recognition network for all classes. It also means we do not need to trade-off prior scores against image residual scores because there is only one image residual.

The ability to extract motor programs could also be used to enhance the training set. [7] shows that error rates can be halved by using smooth vector distortion fields to create extra training data. They argue that these fields simulate "uncontrolled oscillations of the hand muscles dampened by inertia". Motor noise may be better modelled by adding noise to an actual motor program as shown in figure 1. Notice that this produces a wide variety of non-blurry images and it can also change the topology.

The techniques we have used for extracting motor programs from digit images may be applicable to speech. There are excellent generative models that can produce almost perfect speech if they are given the right formant parameters [10]. Using one of these generative models we may be able to train a large number of specialized recognition networks to extract formant parameters from speech without requiring labeled training data. Once this has been done, labeled data would be available for training a single feed-forward network that could recover accurate formant parameters which could be used for real-time recognition.

**Acknowledgements** We thank Steve Isard, David MacKay and Allan Jepson for helpful discussions. This research was funded by NSERC, CFI and OIT. GEH is a fellow of the Canadian Institute for Advanced Research and holds a Canada Research Chair in machine learning.

## Footnotes

[1]We can add all sorts of parameters to the hand-coded generative model and then get the recognition networks to learn to extract the appropriate values for each image. The global mass and viscosity as well as the spacing of the rails that hold the springs can be learned. We can even implement affine-like transformations by attaching the four springs to endpoints whose eight coordinates are given by the recognition networks. These extra parameters make the learning slower and, for the normalized digits, they do not improve discrimination, probably because they help the wrong digit models as much as the right one.

# References

[1] D. M. MacKay. Mindlike behaviour in artefacts. *British Journal for Philosophy of Science*, 2:105–121, 1951.

[2] M. Halle and K. Stevens. Speech recognition: A model and a program for research. *IRE Transactions on Information Theory*, IT-8 (2):155–159, 1962.

[3] Murray Eden. Handwriting and pattern recognition. *IRE Transactions on Information Theory*, IT-8 (2):160–166, 1962.

[4] J.M. Hollerbach. An oscillation theory of handwriting. *Biological Cybernetics*, 39:139–156, 1981.

[5] G. Mayraz and G. E. Hinton. Recognizing hand-written digits using hierarchical products of experts. *IEEE Transactions on Pattern Analysis and Machine Intelligence*, 24:189–197, 2001.

[6] D. Decoste and B. Schoelkopf. Training invariant support vector machines. *Machine Learning*, 46:161–190, 2002.

[7] Patrice Y. Simard, Dave Steinkraus, and John Platt. Best practice for convolutional neural networks applied to visual document analysis. In *International Conference on Document Analysis and Recogntion (ICDAR), IEEE Computer Society, Los Alamitos*, pages 958–962, 2003.

[8] Y. LeCun, L. Bottou, Y. Bengio, and P. Haffner. Gradient-based learning applied to document recognition. *Proceedings of the IEEE*, 86(11):2278–2324, November 1998.

[9] A. Jacobs R. *Increased Rates of Convergence Through Learning Rate Adaptation. Technical Report: UM-CS-1987-117*. University of Massachusetts, Amherst, MA, 1987.

[10] W. Holmes, J. Holmes, and M. Judd. Extension of the bandwith of the jsru parallel-formant synthesizer for high quality synthesis of male and female speech. In *Proceedings of ICASSP 90 (1)*, pages 313–316, 1990.
